# Optimality of Reinforcement Learning Algorithms with Linear Function Approximation

**Ralf Schoknecht**
ILKD
University of Karlsruhe, Germany
*ralf.schoknecht@ilkd.uni-karlsruhe.de*

## Abstract

There are several reinforcement learning algorithms that yield approximate solutions for the problem of policy evaluation when the value function is represented with a linear function approximator. In this paper we show that each of the solutions is optimal with respect to a specific objective function. Moreover, we characterise the different solutions as images of the optimal exact value function under different projection operations. The results presented here will be useful for comparing the algorithms in terms of the error they achieve relative to the error of the optimal approximate solution.

## 1 Introduction

In large domains the determination of an optimal value function via a tabular representation is no longer feasible with respect to time and memory considerations. Therefore, reinforcement learning (RL) algorithms are combined with linear function approximation schemes. However, the different RL algorithms, that all achieve the same optimal solution in the tabular case, converge to different solutions when combined with function approximation. Up to now it is not clear which of the solutions, i.e. which of the algorithms, should be preferred. One reason is that a characterisation of the different solutions in terms of the objective functions they optimise is partly missing. In this paper we state objective functions for the TD(0) algorithm [9], the LSTD algorithm [4, 3] and the residual gradient algorithm [1] applied to the problem of policy evaluation, i.e. the determination of the value function for a fixed policy. Moreover, we characterise the different solutions as images of the optimal exact value function under different projection operations. We think that an analysis of the different optimisation criteria and the projection operations will be useful for determining the errors that the different algorithms achieve relative to the error of the theoretically optimal approximate solution. This will yield a criterion for selecting an *optimal* RL algorithm. For the TD(0) algorithm such error bounds with respect to a specific norm are already known [2, 10] but for the other algorithms there are no comparable results.

## 2 Exact Policy Evaluation

For a Markov decision process (MDP) with finite state space $S$ ($|S| = N$), action space $A$, state transition probabilities $p : (S, S, A) \to [0, 1]$ and stochastic reward function $r : (S, A) \to \mathbb{R}$ policy evaluation is concerned with solving the Bellman equation

$$V^\mu = \gamma P^\mu V^\mu + R^\mu \tag{1}$$

for a fixed policy $\mu : S \to A$. $V_i^\mu$ denotes the value of state $s_i$, $P_{i,j}^\mu = p(s_i, s_j, \mu(s_i))$, $R_i^\mu = E\{r(s_i, \mu(s_i))\}$ and $\gamma$ is the discount factor. As the policy $\mu$ is fixed we will omit it in the following to make notation easier.

The fixed point $V^*$ of equation (1) can be determined iteratively with an operator $T : \mathbb{R}^N \to \mathbb{R}^N$ by

$$TV^n = V^{n+1} = \gamma P V^n + R. \tag{2}$$

This iteration converges to a unique fixed point [2], that is given by

$$V^* = (I - \gamma P)^{-1} R, \tag{3}$$

where $(I - \gamma P)$ is invertible for every stochastic matrix $P$.

## 3 Approximate Policy Evaluation

If the state space $S$ gets too large the exact solution of equation (1) becomes very costly with respect to both memory and computation time. Therefore, often linear feature-based function approximation is applied. The value function $V$ is represented as a linear combination of basis functions $H := \{\Phi_1, \ldots, \Phi_F\}$ which can be written as $V = \Phi w$, where $w \in \mathbb{R}^F$ is the parameter vector describing the linear combination and $\Phi = (\Phi_1 | \ldots | \Phi_F) \in \mathbb{R}^{N \times F}$ is the matrix with the basis functions as columns. The rows of $\Phi$ are the feature vectors $\varphi(s_i) \in \mathbb{R}^F$ for the states $s_i$.

### 3.1 The Optimal Approximate Solution

If the transition probability matrix $P$ were known, then the optimal exact solution $V^* = (I - \gamma P)^{-1} R$ could be computed directly. The optimal approximation to this solution is obtained by minimising $||\Phi w - V^*||$ with respect to $w$. Therefore, a notion of norm must exist. Generally a symmetric positive definite matrix $D$ can be used to define a norm according to $|| \cdot ||_D = \sqrt{\langle \cdot, \cdot \rangle_D}$ with the scalar product $\langle x, y \rangle_D = x^T D y$. The optimal solution that can be achieved with the linear function approximator $\Phi w$ then is the orthogonal projection of $V^*$ onto $[\Phi]$, i.e. the span of the columns of $\Phi$. Let $\Phi$ have full column rank. Then the orthogonal projection on $[\Phi]$ according to the norm $|| \cdot ||_D$ is defined as $\Pi_D = \Phi(\Phi^T D \Phi)^{-1} \Phi^T D$. We denote the optimal approximate solution by $V_D^{SL} = \Pi_D V^*$. The corresponding parameter vector $w_D^{SL}$ with $V_D^{SL} = \Phi w_D^{SL}$ is then given by

$$w_D^{SL} = (\Phi^T D \Phi)^{-1} \Phi^T D V^* = (\Phi^T D \Phi)^{-1} \Phi^T D (I - \gamma P)^{-1} R. \tag{4}$$

Here, SL stands for supervised learning because $w_D^{SL}$ minimises the weighted quadratic error

$$\min_{w \in \mathbb{R}^F} \frac{1}{2} ||\Phi w - V^*||_D^2 = \frac{1}{2} (\Phi w_D^{SL} - V^*)^T D (\Phi w_D^{SL} - V^*) = \frac{1}{2} ||V_D^{SL} - V^*||_D^2 \tag{5}$$

for a given $D$ and $V^*$, which is the objective of a supervised learning method. Note, that $V^*$ equals the expected discounted accumulated reward along a sampled trajectory under the fixed policy $\mu$, i.e. $V^*(s_0) = E[\sum_{t=0}^{\infty} r(s_t, \mu(s_t))]$ for every $s_0 \in S$. These are exactly the samples obtained by the TD(1) algorithm [9]. Thus, the TD(1) solution is equivalent to the optimal approximate solution.

## 3.2 The Iterative TD Algorithm

In the approximate case the Bellman equation (1) becomes

$$\Phi w = \gamma P \Phi w + R \qquad (6)$$

A popular algorithm for updating the parameter vector $w$ after a single transition $x_i \to z_i$ with reward $r_i$ is the stochastic sampling-based TD(0)-algorithm [9]

$$w^{n+1} = w^n + \alpha \varphi(x_i)[r_i + \gamma \varphi(z_i)^T w^n - \varphi(x_i)^T w^n] = (I_F + \alpha A_i)w^n + \alpha b_i, \qquad (7)$$

where $\alpha$ is the learning rate, $A_i = \varphi(x_i)[\gamma \varphi(z_i) - \varphi(x_i)]^T$, $b_i = \varphi(x_i)r_i$ and $I_F$ is the identity matrix in $\mathbb{R}^F$. Let $\rho$ be a probability distribution on the state space $S$. Furthermore, let $x_i$ be sampled according to $\rho$, $z_i$ be sampled according to $p(x_i, \cdot)$ and $r_i$ be sampled according to $r(x_i)$. We will use $E_\rho[\cdot]$ to denote the expectation with respect to the distribution $\rho$. Let $A_{D_\rho}^{TD} = E_\rho[A_i]$ and $b_{D_\rho}^{TD} = E_\rho[b_i]$. If the learning rate decays according to

$$\sum_t \alpha_t = \infty \quad \sum_t \alpha_t^2 < \infty, \qquad (8)$$

then, in the average sense, the stochastic TD(0) algorithm (7) behaves like the deterministic iteration

$$w^{n+1} = (I + \alpha A_{D_\rho}^{TD})w^n + \alpha b_{D_\rho}^{TD}, \qquad (9)$$

with

$$A_{D_\rho}^{TD} = -\Phi^T D_\rho (I - \gamma P)\Phi, \qquad b_{D_\rho}^{TD} = \Phi^T D_\rho R, \qquad (10)$$

where $D_\rho = \mathrm{diag}(\rho)$ is the diagonal matrix with the elements of $\rho$ and $R$ is the vector of expected rewards [2] (Lemma 6.5, Lemma 6.7). In particular the stochastic TD(0) algorithm converges if and only if the deterministic algorithm (9) converges. Furthermore, if both algorithms converge they converge to the same fixed point.

An iteration of the form (9) converges if all eigenvalues of the matrix $I + \alpha A_{D_\rho}^{TD}$ lie within the unit circle [5]. For a matrix $A_{D_\rho}^{TD}$ that has only eigenvalues with negative real part and a learning rate $\alpha_t$ that decays according to (8) there is a $t^*$ such that the eigenvalues of $I + \alpha_t A_{D_\rho}^{TD}$ lie inside the unit circle for all $t > t^*$. Hence, for a decaying learning rate the deterministic TD(0) algorithm converges if all eigenvalues of $A_{D_\rho}^{TD}$ have a negative real part. Since this requirement is not always fulfilled the TD algorithm possibly diverges as shown in [1]. This divergence is due to the positive eigenvalues of $A_{D_\rho}^{TD}$ [8].

However, under special assumptions convergence of the TD(0) algorithm can be shown [2]. Let the feature matrix $\Phi \in \mathbb{R}^{N \times F}$ have full rank, where $F \leq N$, i.e. there are not more parameters than states). This results in no loss of generality because the linearly dependent columns of $\Phi$ can be eliminated without changing the power of the approximation architecture. The most important assumption concerns the sampling of the states that is reflected in the matrix $D$. Let the Markov chain be aperiodic and recurrent. Besides the aperiodicity requirement, this assumption results in no loss of generality because transient states can be eliminated. Then a steady-state distribution $\pi$ of the Markov chain exists. When sampling the states according to this steady-state distribution, i.e. $D = D_\pi = \mathrm{diag}(\pi)$, it can be shown that $A_{D_\pi}^{TD}$ is negative definite [2] (Lemma 6.6). This immediately yields that all eigenvalues are negative which in turn yields convergence of the TD(0) algorithm with decaying learning rate.

In the next section we will characterise the limit value $V_{D_\pi}^{TD}$ as the projection of $V^*$ in a more general setting. However, for the sampling distribution $\pi$ there is another interesting interpretation of $V_{D_\pi}^{TD}$ as the fixed point of $\Pi_{D_\pi} T$, where $\Pi_{D_\pi}$ is the orthogonal projection with respect to $D_\pi$ onto $[\Phi]$, as defined in section 3.1, and $T$ is the update operator defined in (2) [2, 10]. In the following we use this fact to deduce a new formula for $V_{D_\pi}^{TD}$ that has a form similar to $V^*$ in (3). Before we proceed, we need the following lemma

**Lemma 1** *The matrix $I - \gamma \Pi_{D_\pi} P$ is regular.*

**Proof:** The matrix $I - \gamma \Pi_{D_\pi} P$ is regular if and only if it does not have eigenvalue zero. An equivalent condition is that one is not an eigenvalue of $\gamma \Pi_{D_\pi} P$. Therefore, it is sufficient to show that the spectral radius satisfies $\varrho(\gamma \Pi_{D_\pi} P) < 1$. For any matrix norm $|| \cdot ||$ it holds that $\varrho(A) \le ||A||$ [5]. Therefore, we know that $\varrho(\gamma \Pi_{D_\pi} P) \le ||\gamma \Pi_{D_\pi} P||_{D_\pi}$, where the vector norm $|| \cdot ||_{D_\pi}$ induces the matrix norm $|| \cdot ||_{D_\pi}$ by the standard definition $||A||_{D_\pi} = \sup_{||x||_{D_\pi}=1}\{||Ax||_{D_\pi}\}$. With this definition and with the fact that $||Px||_{D_\pi} \le ||x||_{D_\pi}$ for all $x$ [2] (Lemma 6.4) we obtain $||P||_{D_\pi} = \sup_{||x||_{D_\pi}=1}\{||Px||_{D_\pi}\} \le \sup_{||x||_{D_\pi}=1}\{||x||_{D_\pi}\} = 1$. Moreover, we have $||\Pi_{D_\pi}||_{D_\pi} = \sup_{||x||_{D_\pi}=1}\{||\Pi_{D_\pi} x||_{D_\pi}\} \le \sup_{||x||_{D_\pi}=1}\{||x||_{D_\pi}\} = 1$, where we used the well known fact that an orthogonal projection $\Pi_{D_\pi}$ is a non-expansion with respect to the vector norm $|| \cdot ||_{D_\pi}$. Putting all together we obtain $\varrho(\gamma \Pi_{D_\pi} P) \le ||\gamma \Pi_{D_\pi} P||_{D_\pi} \le \gamma ||\Pi_{D_\pi}||_{D_\pi} \cdot ||P||_{D_\pi} \le \gamma < 1$. $\square$

We can now solve the fixed point equation $V_{D_\pi}^{TD} = \Pi_{D_\pi} T V_{D_\pi}^{TD}$ and obtain

$$V_{D_\pi}^{TD} = (I - \gamma \Pi_{D_\pi} P)^{-1} \Pi_{D_\pi} R = (I - \gamma \widetilde{P})^{-1} \widetilde{R}, \tag{11}$$

with $\widetilde{P} = \Pi_{D_\pi} P$ and $\widetilde{R} = \Pi_{D_\pi} R$. This resembles equation (3) for the exact solution of the policy evaluation problem. The TD(0) solution with sampling distribution $\pi$ can thus be interpreted as exact solution of the "projected" policy evaluation problem with $\widetilde{P}$ and $\widetilde{R}$. Note, that compared to the TD(1) solution of the approximate policy evaluation problem $V_{D_\pi}^{SL} = \Pi_{D_\pi}(I - \gamma P)^{-1} R$ with weighting matrix $D_\pi$ equation (11) only differs in the position of the projection operator. This leads to an interesting comparison of TD(0) and TD(1). While TD(0) yields the exact solution of the projected problem, TD(1) yields the projected solution of the exact problem.

### 3.3 The Least-Squares TD Algorithm

Besides the iterative solution of (6) often a direct solution by matrix inversion is computed using equation (9) in the fixed point form $A_{D_\rho}^{TD} w_{D_\rho}^{TD} + b_{D_\rho}^{TD} = 0$. This approach is known as least-squares TD (LSTD) [4, 3]. It is only required that $A_{D_\rho}^{TD}$ be invertible, i.e. that the eigenvalues be unequal zero. In contrast to the iterative TD algorithm the eigenvalues need not have negative real parts. Therefore, LSTD offers the possibility of using sampling distributions $\rho$ other than the steady-state distribution $\pi$ [6, 7] Thus, parts of the state space that would be rarely visited under the steady-state distribution can now be visited more frequently which makes the approximation of the value function more reliable. This is necessary if the result of policy evaluation should be used in a policy improvement step because otherwise the action choice in rarely visited states may be bad [6].

For the following let the feature matrix have full column rank. As described above this results in no loss of generality. LSTD allows to sample the states with an arbitrary sampling distribution $\rho$. If there are states $s$ that are not visited under $\rho$,

i.e. $\rho(s) = 0$, then these states can be eliminated from the Markov chain. Hence, without loss of generality we assume that the matrix $D_\rho = \text{diag}(\rho)$ is invertible. These conditions ensure the invertibility of $A_{D_\rho}^{TD}$ and according to [4, 3] the LSTD solution is given by

$$w_{D_\rho}^{TD} = (-A_{D_\rho}^{TD})^{-1} b_{D_\rho}^{TD}. \tag{12}$$

Note, that the matrix $A_{D_\rho}^{TD}$ and the vector $b_{D_\rho}^{TD}$ can be computed from samples such that the model $P$ does not need to be known. Note also that in general $w_{D_\rho}^{TD} \neq w_{D_\rho}^{SL}$ as discussed in [3]. This means, that the TD(0) solution $w_{D_\rho}^{TD}$ and the TD(1) solution $w_{D_\rho}^{SL}$ may differ when function approximation is used.

Depending on the sampling distribution $\rho$ the LSTD approach may be the only way of computing the fixed point of (9) because the corresponding iterative TD(0) algorithm may diverge due to positive eigenvalues. However, if the TD(0) algorithm converges the limit coincides with the LSTD solution $w_{D_\rho}^{TD}$.

For the value function $V_{D_\rho}^{TD}$ achieved by the LSTD algorithm the following holds

$$V_{D_\rho}^{TD} \;=\; \Phi w_{D_\rho}^{TD} \overset{(12)}{=} \Phi(-A_{D_\rho}^{TD})^{-1} b_{D_\rho}^{TD} = \Phi \left[ (-A_{D_\rho}^{TD})^T (-A_{D_\rho}^{TD}) \right]^{-1} (-A_{D_\rho}^{TD})^T b_{D_\rho}^{TD}$$

$$\overset{(3),(10)}{=} \Pi_{(I-\gamma P)^T D_\rho^T \Phi \Phi^T D_\rho (I-\gamma P)} V^* = \Pi_{D_\rho^{TD}} V^*. \tag{13}$$

We define $D_\rho^{TD} = (I-\gamma P)^T D_\rho^T \Phi \Phi^T D_\rho (I-\gamma P)$. As $\Phi\Phi^T$ is singular in general, the matrix $D_\rho^{TD}$ is symmetric and positive semi-definite. Hence, it defines a semi-norm $\| \cdot \|_{D_\rho^{TD}}$. Thus, the LSTD solution is obtained by projecting $V^*$ onto $[\Phi]$ with respect to $\| \cdot \|_{D_\rho^{TD}}$. After having deduced this new relation between the optimal solution $V^*$ and $V_{D_\rho}^{TD}$ we can characterise $w_{D_\rho}^{TD}$ as minimising the corresponding quadratic objective function.

$$\min_{c \in \mathbb{R}^F} \frac{1}{2} \| \Phi w - V^* \|_{D_\rho^{TD}}^2 = \frac{1}{2} (\Phi w_{D_\rho}^{TD} - V^*)^T D_\rho^{TD} (\Phi w_{D_\rho}^{TD} - V^*) = \frac{1}{2} \| V_{D_\rho}^{TD} - V^* \|_{D_\rho^{TD}}^2. \tag{14}$$

It can be shown that the value of the objective function for the LSTD solution is zero, i.e. $\| V_{D_\rho}^{TD} - V^* \|_{D_\rho^{TD}}^2 = 0$. With equation (14) we have shown that the LSTD solution minimises a certain error metric. The form of this error metric is similar to (5). The only difference lies in the norm that is used. This unifies the characterisation of the solutions that are achieved by different algorithms.

## 3.4 The Residual Gradient Algorithm

There is a third approach to solving equation (6). The residual gradient algorithm [1] directly minimises the weighted Bellman error

$$\frac{1}{2} \| (I - \gamma P) \Phi w - R \|_{D_\rho}^2 \tag{15}$$

by gradient descent. The resulting update rule of the deterministic algorithm has a form similar to (9)

$$w^{n+1} = (I + \alpha A_{D_\rho}^{RG}) w^n + \alpha b_{D_\rho}^{RG}, \tag{16}$$

with

$$A_{D_\rho}^{RG} = -\Phi^T (I - \gamma P)^T D_\rho (I - \gamma P) \Phi, \qquad b_{D_\rho}^{RG} = \Phi^T (I - \gamma P^T) D_\rho R, \tag{17}$$

where $D_\rho$ is again the diagonal matrix with the visitation probabilities $\rho_i$ on its diagonal. As all entries on the diagonal are nonnegative, $D_\rho$ can be decomposed

into $\sqrt{D_\rho}^T\sqrt{D_\rho}$. Hence, we can write $A_{D_\rho}^{RG} = -(\sqrt{D_\rho}(I-\gamma P)\Phi)^T\sqrt{D_\rho}(I-\gamma P)\Phi$. Therefore, $A_{D_\rho}^{RG}$ is negative semidefinite. If $\Phi$ has full column rank and $D_\rho$ is regular, i.e. the visitation probability for every state is positive, then $A_{D_\rho}^{RG}$ is negative definite. Therefore, all eigenvalues of $A_{D_\rho}^{RG}$ are negative, which yields convergence of the residual gradient algorithm (16) for a decaying learning rate independently of the weighting $D_\rho$, the function approximator $\Phi$ and the transition probabilities $P$. The equivalence of the limit value of the deterministic and the stochastic version of the residual gradient algorithm can be proven with an argument similar to that in [2] for the equivalence of the deterministic and the stochastic version of the TD(0) algorithm in equations (7) and (9) respectively. Note also that the matrix $A_{D_\rho}^{RG}$ and the vector $b_{D_\rho}^{RG}$ can be computed from samples so that the model $P$ does not need to be known for the deterministic residual gradient algorithm.

If $A_{D_\rho}^{RG}$ is invertible a unique limit of the iteration (16) exists. It can be directly computed via the fixed point form, which yields the new identity

$$w_{D_\rho}^{RG} = (-A_{D_\rho}^{RG})^{-1}b_{D_\rho}^{RG} = (\Phi^T(I-\gamma P)^T D_\rho(I-\gamma P)\Phi)^{-1}\Phi^T(I-\gamma P)^T D_\rho R. \quad (18)$$

This solution of the residual gradient algorithm is related to the optimal solution (4) of the approximate Bellman equation (6) as described in the following lemma.

**Lemma 2** *The solution $w_{D_\rho}^{RG}$ of the residual gradient algorithm with weighting matrix $D_\rho$ is equivalent to the optimal supervised learning solution $w_{D_\rho^{RG}}^{SL}$ of the approximate Bellman equation (6) with weighting matrix $D_\rho^{RG} = (I-\gamma P)^T D_\rho(I-\gamma P)$.*

**Proof:**

$$w_{D_\rho}^{RG} = (\Phi^T(I-\gamma P)^T D_\rho(I-\gamma P)\Phi)^{-1}\Phi^T(I-\gamma P)^T D_\rho R$$
$$= (\Phi^T D_\rho^{RG}\Phi)^{-1}\Phi^T(I-\gamma P)^T D_\rho(I-\gamma P)(I-\gamma P)^{-1}R$$
$$= (\Phi^T D_\rho^{RG}\Phi)^{-1}\Phi^T D_\rho^{RG}V^* = w_{D_\rho^{RG}}^{SL},$$

where we used the fact that $V^* = (I-\gamma P)^{-1}R$. $\qquad\square$

Therefore, $w_{D_\rho}^{RG}$ can be interpreted as the orthogonal projection of the optimal solution $V^*$ onto $[\Phi]$ with respect to the scalar product defined by $D_\rho^{RG}$. This yields a new equivalent formula for the Bellman error (15)

$$\frac{1}{2}\|(I-\gamma P)\Phi w - R\|_{D_\rho}^2 = \frac{1}{2}((I-\gamma P)\Phi w - R)^T D_\rho((I-\gamma P)\Phi w - R)$$
$$= \frac{1}{2}(\Phi w - V^*)^T(I-\gamma P)^T D_\rho(I-\gamma P)(\Phi w - V^*) = \frac{1}{2}\|\Phi w - V^*\|_{D_\rho^{RG}}^2. \quad (19)$$

The Bellman error is the objective function that is minimised by the residual gradient algorithm. As we have just shown, this objective function can be expressed in a form similar to (5), where the only difference lies in the norm that is used. Thus, we have shown that the solution of the residual gradient algorithm can also be characterised in the general framework of quadratic error metrics $\|\Phi w - V^*\|_D$. As a direct consequence we can represent the solution as an orthogonal projection $V_{D_\rho}^{RG} = \Phi w_{D_\rho}^{RG} = \Pi_{D_\rho^{RG}}V^*$.

According to section 3.2 an iteration of the form (16) generally converges for matrices $A$ with eigenvalues that have negative real parts. However, the fact that $A_{D_\rho}^{RG}$ is symmetric assures convergence even for singular $A_{D_\rho}^{RG}$ [8] (Proposition 1). Thus,

Table 1: Overview over the solutions of different RL algorithms. The supervised learning (SL) approach, the TD(0) algorithm, the LSTD algorithm and the residual gradient (RG) algorithm are analysed in terms of the conditions of solvability. Moreover, we summarise the optimisation criteria that the different algorithms minimise and characterise the different solutions in terms of the projection of the optimal solution $V^*$ onto $[\Phi]$. If the visitation distribution is arbitrary, we write $\forall \rho$.

| | | SL | TD | LSTD | RG |
|---|---|---|---|---|---|
| solvability: | condition for $\lambda_i$ | $-$ | $\mathrm{Re}(\lambda_i) < 0$ | $\lambda_i \neq 0$ | $\mathrm{Re}(\lambda_i) \leq 0$ |
| | condition for $\rho$ | $\forall \rho$ | $\rho = \pi$ | $\rho(s) \neq 0$ | $\forall \rho$ |
| optimisation criterion | | eq. (5) | eq. (14) | eq. (14) | eq. (19) |
| characterisation as projection | | $\Pi_{D_\rho} V^*$ | $\Pi_{D_\pi^{TD}} V^*$ | $\Pi_{D_\rho^{TD}} V^*$ | $\Pi_{D_\rho^{RG}} V^*$ |

the residual gradient algorithm (16) converges for any matrix $A_{D_\rho}^{RG}$ that is of the form (17) and in case $A_{D_\rho}^{RG}$ is regular the limit is given by (18). Note that a matrix $\Phi$ which does not have full column rank leads to ambiguous solutions $w_{D_\rho}^{RG}$ that depend on the initial value $w^0$. However, the corresponding $V_{D_\rho}^{RG} = \Phi w_{D_\rho}^{RG}$ are the same. For singular $D_\rho$ the matrix $D_\rho^{RG} = (I - \gamma P)^T D_\rho (I - \gamma P)$ is also singular. Thus, the limit $V_{D_\rho}^{RG}$ may not be unique but may depend itself on the initial value $w^0$. The reason is that there may be a whole subspace of $[\Phi]$ with dimension larger than zero that minimises $||V_{D_\rho}^{RG} - V^*||_{D^{RG}}$ because $||\cdot||_{D^{RG}}$ is now only a semi-norm. But for all minimising $V_{D_\rho}^{RG}$ the Bellman error is the same, i.e. with respect to the Bellman error all the solutions $V_{D_\rho}^{RG}$ are equivalent [8] (Proposition 1).

### 3.5 Synopsis of the Different Solutions

In Table 1 we give a brief overview of the solutions that the different RL algorithms yield. An SL solution can be computed for arbitrary weighting matrices $D_\rho$ induced by a sampling distribution $\rho$. For the three RL algorithms (TD, LSTD, RG) solvability conditions can be either formulated in terms of the eigenvalues of the iteration matrix $A$ or in terms of the sampling distribution $\rho$. The iterative TD(0) algorithm has the most restrictive conditions for solvability both for the eigenvalues of the iteration matrix $A$, whose real parts must be smaller than zero, and for the sampling distribution $\rho$, which must equal the steady-state distribution $\pi$. The LSTD method only requires invertibility of $A_{D_\rho}^{TD}$. This is satisfied if $\Phi$ has full column rank and if the visitation distribution $\rho$ samples every state $s$ infinitely often, i.e. $\rho(s) \neq 0$ for all $s \in S$. In contrast to that the residual gradient algorithm converges independently of $\rho$ and the concrete $A_{D_\rho}^{RG}$ because all these matrices have eigenvalues with nonpositive real parts.

All solutions can be characterised as minimising a quadratic optimisation criterion $||\Phi w - V^*||_D$ with corresponding matrix $D$. The SL solution optimises the weighted quadratic error (5), RG optimises the weighted Bellman error (19) and both TD and LSTD optimise the quadratic function (14) with weighting matrices $D_\pi^{TD}$ and $D_\rho^{TD}$ respectively. With the assumption of regular $D_\rho$, i.e. $\rho(s) \neq 0$ for all $s \in S$, the solutions $V$ can be characterised as images of the optimal solution $V^*$ under different orthogonal projections (optimal, RG) and projections that minimise a semi-norm (TD, LSTD). For singular $D_\rho$ see the remarks on ambiguous solutions in section 3.4.

Let us finally discuss the case of a quasi-tabular representation of the value function that is obtained for regular $\Phi$ and let all states be visited infinitely often, i.e. $D_\rho$ is regular. Due to the invertibility of $\Phi$ we have $[\Phi] = \mathbb{R}^N$. Thus, the optimal solution $V^*$ is exactly representable because $V^* \in [\Phi]$. Moreover, every projection operator $\Pi : \mathbb{R}^N \to [\Phi]$ reduces to the identity. Therefore, all the projection operators for the different algorithms are equivalent to the identity. Hence, with a quasi-tabular representation all the algorithms converge to the optimal solution $V^*$.

## 4 Conclusions

We have presented an analysis of the solutions that are achieved by different reinforcement learning algorithms combined with linear function approximation. The solutions of all the examined algorithms, TD(0), LSTD and the residual gradient algorithm, can be characterised as minimising different corresponding quadratic objective function. As a consequence, each of the value functions, that one of the above algorithms converges to, can be interpreted as image of the optimal exact value function under a corresponding orthogonal projection. In this general framework we have given the first characterisation of the approximate TD(0) solution in terms of the minimisation of a quadratic objective function. This approach allows to view the TD(0) solution as exact solution of a projected learning problem. Moreover, we have shown that the residual gradient solution and the optimal approximate solution only differ in the weighting of the error between the exact and the approximate solution. In future research we intend to use the results presented here for determining the errors of the different solutions relative to the optimal approximate solution with respect to a given norm. This will yield a criterion for selecting reinforcement learning algorithms that achieve optimal solution quality.

## References

[1] L. C. Baird. Residual algorithms: Reinforcement learning with function approximation. *Proc. of the Twelfth International Conference on Machine Learning*, 1995.

[2] D. P. Bertsekas and J. N. Tsitsiklis. *Neuro Dynamic Programming*. Athena Scientific, Belmont, Massachusetts, 1996.

[3] J.A. Boyan. Least-squares temporal difference learning. In *Proceeding of the Sixteenth International Conference on Machine Learning*, pages 49–56, 1999.

[4] S.J Bradtke and A.G. Barto. Linear least-squares algorithms for temporal difference learning. *Machine Learning*, 22:33–57, 1996.

[5] A. Greenbaum. *Iterative Methods for Solving Linear Systems*. SIAM, 1997.

[6] D. Koller and R. Parr. Policy iteration for factored mdps. In *Proc. of the Sixteenth Conference on Uncertainty in Artificial Intelligence (UAI)*, pages 326–334, 2000.

[7] M. G. Lagoudakis and R. Parr. Model-free least-squares policy iteration. In *Advances in Neural Information Processing Systems*, volume 14, 2002.

[8] R. Schoknecht and A. Merke. Convergent combinations of reinforcement learning with function approximation. In *Advances in Neural Information Processing Systems*, volume 15, 2003.

[9] R. S. Sutton. Learning to predict by the methods of temporal differences. *Machine Learning*, 3:9–44, 1988.

[10] J. N. Tsitsiklis and B. Van Roy. An analysis of temporal-difference learning with function approximation. *IEEE Transactions on Automatic Control*, 1997.
